# Optimal Sampling of Natural Images: A Design Principle for the Visual System?

William Bialek,[a,b] Daniel L. Ruderman,[a] and A. Zee[c]

[a]Department of Physics, and
Department of Molecular and Cell Biology
University of California at Berkeley
Berkeley, California 94720

[b]NEC Research Institute
4 Independence Way
Princeton, New Jersey 08540

[c]Institute for Theoretical Physics
University of California at Santa Barbara
Santa Barbara, California 93106

## Abstract

We formulate the problem of optimizing the sampling of natural images using an array of linear filters. Optimization of information capacity is constrained by the noise levels of the individual channels and by a penalty for the construction of long-range interconnections in the array. At low signal-to-noise ratios the optimal filter characteristics correspond to bound states of a Schrödinger equation in which the signal spectrum plays the role of the potential. The resulting optimal filters are remarkably similar to those observed in the mammalian visual cortex and the retinal ganglion cells of lower vertebrates. The observed scale invariance of natural images plays an essential role in this construction.

# 1 Introduction

Under certain conditions the visual system is capable of performing extremely efficient signal processing [1]. One of the major theoretical issues in neural computation is to understand how this efficiency is reached given the constraints imposed by the biological hardware. Part of the problem [2] is simply to give an informative representation of the visual world using a limited number of neurons, each of which has a limited information capacity. The information capacity of the visual system is determined in part by the spatial transfer characteristics, or "receptive fields," of the individual cells. From a theoretical point of view we can ask if there exists an optimal choice for these receptive fields, a choice which maximizes the information transfer through the system given the hardware constraints. We show that this optimization problem has a simple formulation which allows us to use the intuition developed through the variational approach to quantum mechanics.

In general our approach leads to receptive fields which are quite unlike those observed for cells in the visual cortex. In particular orientation selectivity is not a generic prediction. The optimal filters, however, depend on the statistical properties of the images we are trying to sample. Natural images have a symmetry — scale invariance [4] — which saves the theory: The optimal receptive fields for sampling of *natural* images are indeed orientation selective and bear a striking resemblance to observed receptive field characteristics in the mammalian visual cortex as well as the retinal ganglion of lower vertebrates.

# 2 General Theoretical Formulation

We assume that images are defined by a scalar field $\phi(\mathbf{x})$ on a two dimensional surface with coordinates $\mathbf{x}$. This image is sampled by an array of cells whose outputs $Y_n$ are given by

$$Y_n = \int d^2x\, F(\mathbf{x} - \mathbf{x}_n)\phi(\mathbf{x}) + \eta_n, \tag{1}$$

where the cell is loacted at site $\mathbf{x}_n$, its spatial transfer function or receptive field is defined by $F$, and $\eta$ is an independent noise source at each sampling point. We will assume for simplicity that the noise source is Gaussian, with $\langle \eta^2 \rangle = \sigma^2$. Our task is to find the receptive field $F$ which maximizes the information provided about $\phi$ by the set of outputs $\{Y_n\}$.

If the field $\phi$ is itself chosen from a stationary Gaussian distribution then the information carried by the $\{Y_n\}$ is given by [3]

$$I = \frac{1}{2\ln 2}\operatorname{Tr}\ln\left[\delta_{nm} + \frac{1}{\sigma^2}\int \frac{d^2k}{(2\pi)^2}e^{i\mathbf{k}\cdot(\mathbf{x}_n-\mathbf{x}_m)}|\tilde{F}(\mathbf{k})|^2 S(\mathbf{k})\right], \tag{2}$$

where $S(\mathbf{k})$ is the power spectrum of the signals,

$$S(\mathbf{k}) = \int d^2y\, e^{-i\mathbf{k}\cdot\mathbf{y}}\langle \phi(\mathbf{x}+\mathbf{y})\phi(\mathbf{x})\rangle, \tag{3}$$

and $\tilde{F}(\mathbf{k}) = \int d^2x\, e^{-i\mathbf{k}\cdot\mathbf{x}}F(\mathbf{x})$ is the receptive field in momentum (Fourier) space.

At low signal-to-noise ratios (large $\sigma^2$) we have

$$I \approx \frac{N}{2\ln 2\sigma^2} \int \frac{d^2k}{(2\pi)^2} |\tilde{F}(\mathbf{k})|^2 S(\mathbf{k}), \tag{4}$$

where $N$ is the total number of cells.

To make our definition of the noise level $\sigma$ meaningful we must constrain the total "gain" of the filters $F$. One simple approach is to normalize the functions $F$ in the usual $L^2$ sense,

$$\int d^2x F^2(\mathbf{x}) = \int \frac{d^2k}{(2\pi)^2} |\tilde{F}(\mathbf{k})|^2 = 1. \tag{5}$$

If we imagine driving the system with spectrally white images, this condition fixes the total signal power passing through the filter.

Even with normalization, optimization of information capacity is still not well-posed. To avoid pathologies we must constrain the scale of variations in $\mathbf{k}$−space. This makes sense biologically since we know that sharp features in $\mathbf{k}$−space can be achieved only by introducing long-range interactions in real space, and cells in the visual system typically have rather local interconnections. We implement this constraint by introducing a penalty proportional to the mean square spatial extent of the receptive field,

$$\int d^2x \, x^2 F^2(x) = \int \frac{d^2k}{(2\pi)^2} |\nabla_k \tilde{F}(\mathbf{k})|^2. \tag{6}$$

With all the constraints we find that, at low signal to noise ratio, our optimization problem becomes that of minimizing the functional

$$C[\tilde{F}] = (1/2)\alpha \int \frac{d^2k}{(2\pi)^2} |\nabla_k \tilde{F}(\mathbf{k})|^2 \quad - \quad \frac{1}{2\ln 2\sigma^2} \int \frac{d^2k}{(2\pi)^2} |\tilde{F}(\mathbf{k})|^2 S(\mathbf{k})$$
$$- \quad \Lambda \int \frac{d^2k}{(2\pi)^2} |\tilde{F}(\mathbf{k})|^2, \tag{7}$$

where $\Lambda$ is a Lagrange multiplier and $\alpha$ measures the strength of the locality constraint. The optimal filters are then solutions of the variational equation,

$$-\frac{\alpha}{2} \nabla_k^2 \tilde{F}(\mathbf{k}) - \frac{1}{2\ln 2\sigma^2} S(\mathbf{k})\tilde{F}(\mathbf{k}) = \Lambda \tilde{F}(\mathbf{k}). \tag{8}$$

We recognize this as the Schrödinger equation for a particle moving in k-space, in which the mass $M = \hbar^2/\alpha$, the potential $V(\mathbf{k}) = -S(\mathbf{k})/2\ln 2\sigma^2$, and $\Lambda$ is the energy eigenvalue. Since we are interested in normalizable $F$, we are restricted to bound states, and the optimal filter is just the bound state wave function.

There are in general several optimal filters, corresponding to the different bound states. Each of these filters gives the same value for the total cost function $C[\tilde{F}]$ and hence is equally "good" in this context. Thus each sampling point should be served by a set of filters rather than just one. Indeed, in the visual cortex one finds a given region of the visual field being sampled by many cells with different spatial frequency and orientation selectivities.

## 3    A Near-Fatal Flaw and its Resolution

If the signal spectra $S(\mathbf{k})$ are isotropic, so that features appear at all orientations across the visual field, all of the bound states of the corresponding Schrödinger equation are eigenstates of angular momentum. But real visual neurons have receptive fields with a single optimal orientation, not the multiple optima expected if the filters $F$ correspond to angular momentum eigenstates. One would like to combine different angular momentum eigenfunctions to generate filters which respond to localized regions of orientation. In general, however, the different angular momenta are associated with different energy eigenvalues and hence it is impossible to form linear combinations which are still solutions of the variational problem.

We *can* construct receptive fields which are localized in orientation if there is some extra symmetry or accidental degeneracy which allows the existence of equal-energy states with different angular momenta. If we believe that real receptive fields are the solutions of our variational problem, it must be the case that the signal spectrum $S(\mathbf{k})$ for natural images possesses such a symmetry.

Recently Field [4] has measured the power spectra of several natural scenes. As one might expect from discussions of "fractal" landscapes, these spectra are scale invariant, with $S(\mathbf{k}) = A/|\mathbf{k}|^2$. It is easy to see that the corresponding quantum mechanics problem is a bit sick — the energy is not bounded from below. In the present context, however, this sickness is a saving grace. The equivalent Schrödinger equation is

$$-\frac{\alpha}{2}\nabla_k^2 \tilde{F}(\mathbf{k}) - \frac{A}{2\ln 2\sigma^2|\mathbf{k}|^2}\tilde{F}(\mathbf{k}) = \Lambda\tilde{F}(\mathbf{k}). \qquad (9)$$

If we take $\mathbf{q} = (\sqrt{2|\Lambda|/\alpha})\mathbf{k}$, then for bound states ($\Lambda < 0$) we find

$$\nabla_q^2 \tilde{F}(\mathbf{q}) + \frac{B}{|\mathbf{q}|^2}\tilde{F}(\mathbf{q}) = \tilde{F}(\mathbf{q}), \qquad (10)$$

with $B = A/\ln 2\sigma^2$. Thus we see that the energy $\Lambda$ can be scaled away; there is no quantization condition. We are free to choose any value of $\Lambda$, but for each such value there are several angular momentum states. Since they correspond to the same energy, superpositions of these states are also solutions of the original variational problem. The scale invariance of natural images is the symmetry we need in order to form localized receptive fields.

## 4    Predicting Receptive Fields

To solve Eq. (9) we find it easier to transform back to real space. The result is

$$r^2(1+r^2)\frac{\partial^2 F}{\partial r^2} + r(1+5r^2)\frac{\partial F}{\partial r} + [r^2(4+B+\partial^2/\partial\phi^2) + \partial^2/\partial\phi^2]F = 0, \qquad (11)$$

where $\phi$ is the angular variable and $r = (\sqrt{\alpha/2|\Lambda|})|\mathbf{x}|$. Angular momentum states $F_m \sim e^{im\phi}$ have the asymptotic $F_m(r \ll 1) \sim r^{\pm m}$, $F_m(r \gg 1) \sim r^{\lambda_\pm(m)}$, with $\lambda_\pm(m) = -2\pm\sqrt{m^2 - B}$. We see that for $m^2 < B$ the solutions are oscillatory functions of $r$, since $\lambda$ has an imaginary part. For $m^2 > B + 4$ the solution can diverge

as $r$ becomes large, and in this case we must be careful to choose solutions which are regular both at the origin and at infinity if we are to maintain the constraint in Eq. (5). Numerically we find that there are no such solutions; the functions which behave as $r^{+|m|}$ near the origin diverge at large $r$ if $m^2 > B + 4$. We conclude that for a given value of $B$, which measures the signal-to-noise ratio, there exists a finite set of angular momentum states; these states can then be superposed to give receptive fields with localized angular sensitivity.

In fact *all* linear combinations of $m$−states are solutions to the variational problem at low signal to noise ratio, so the precise form of orientation tuning is not determined. If we continue our expansion of the information capacity in powers of the signal-to-noise ratio we find terms which will select different linear combinations of the $m$−states and hence determine the precise orientation selectivity. These higher-order terms, however, involve multi-point correlation functions of the image. At the lowest SNR, corresponding to the first term in our expansion, we are sensitive only to the two-point function (power spectrum) of the signal ensemble, which carries no information about angular correlations. A truly predictive theory of orientation tuning must thus rest on measurements of angular correlations in natural images; as far as we know such measurements have not been reported.

Even without knowing the details of the higher-order correlation functions we can make some progress. To begin, it is clear that at very small $B$ orientation selectivity is impossible since there are only $m = 0$ solutions. This is the limit of very low SNR, or equivalently very strong constraints on the locality of the receptive field (large $\alpha$ above). The circularly symmetric receptive fields that one finds in this limit are center-surround in structure, with the surround becoming more prominent as the signal-to-noise ratio is increased. These predictions are in qualitative accord with what one sees in the mammalian retina, which is indeed extremely local — receptive field centers for foveal ganglion cells may consist of just a single cone photoreceptor. As one proceeds to the the cortex the constraints of locality are weaker and orientation selectivity becomes possible. Similarly in lower vertebrates there is a greater range of lateral connectivity in the retina itself, and hence orientation selectivity is possible at the level of the ganglion cell.

To proceed further we have explored the types of receptive fields which can be produced by superposing $m$−states at a given value of $B$. We consider for the moment only even-symmetric receptive fields, so we add all terms in phase. One such receptive field is shown in Fig. 1, together with experimental results for a simple cell in the primary visual cortex of monkeys [5]. It is clear that we can obtain reasonable correspondence between theory and experiment. Obviously we have made no detailed "fit" to the data, and indeed we are just beginning a quantitative comparison of theory with experiment. Much of the arbitrariness in the construction of Fig. 1 will be removed once we have control over the higher terms in the SNR expansion, as described above.

It is interesting that, at low SNR, there is no preferred value for the length scale. Thus the optimal system may choose to sample images at many different scales and at different scales in different regions of the image. The experimental variability in spatial frequency tuning from cell to cell may thus not represent biological sloppiness but rather the fact that any peak spatial frequency constitutes an optimal filter in the sense defined here.

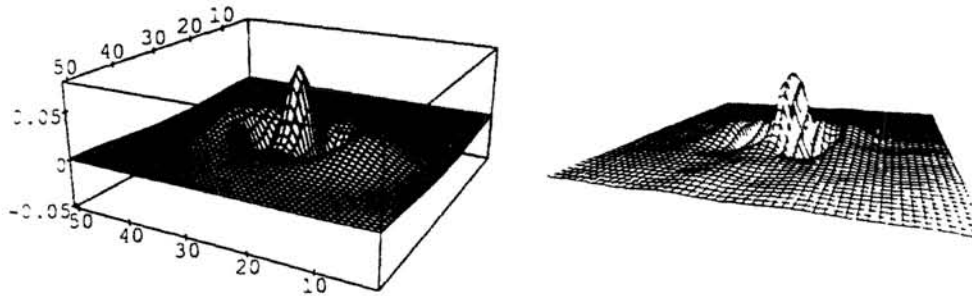

Figure 1: Model (left) and monkey (right) receptive fields. Monkey RF is from reference [5].

## 5   Discussion

The selectivity of cortical neurons for orientation and spatial frequency are among the best known facts about the visual system. Not surprisingly there have been many attempts to derive these features from some theoretical perspective. One approach is to argue that such selectivity provides a natural preprocessing stage for more complex computations. A very different view is that the observed organization of the cortex is a consequence of developmental rules, but this approach does not address the computational function which may be expressed by cortical organization. Finally several authors have considered the possibility that cortical receptive fields are in some sense optimal, so that they can be predicted from a variational principle [6, 7, 8]. Clearly we have adopted this last hypothesis; the issue is whether one can make a compelling argument for any particular variational principle.

Optimization of information capacity seems like a very natural principle to apply in the early stages of visual processing. As we have emphasized, this principle must be supplemented by a knowledge of hardware constraints and of image statistics. Different authors have made different choices, especially for the constraints. Different formulations, however, may be related — optimization of information transfer at some fixed "gain" of the receptive fields is equivalent, through a Legendre transformation, to minimization of the redundancy at fixed information transfer, a problem discussed by Atick and Redlich [8]. This latter approach has given very successful predictions for the structure of ganglion cell receptive fields in cat and monkey, although there are still some arbitrary parameters to be determined. It is our hope that these ideas of receptive fields as solutions to variational problems can be given

more detailed tests in the lower vertebrate retinas, where it is possible to characterize signals and noise at each of three layers of processing cicuitry.

As far as we know our work is unique in that the statistics of natural images, is an essential component of the theory. Indeed the scale invariance of natural images plays a decisive role in our prediction of orientation selectivity; other classes of signals would result in qualitatively different receptive fields. We find this direct linkage between the properties of natural images and the architecture of natural computing systems to be extremely attractive. The semi-quantitative correspondence between predicted and observed receptive fields (Fig. 1) suggests that we have the kernel of a truly predictive theory for visual processing.

## Acknowledgements

We thank K. DeValois, R. DeValois, J. D. Jackson, and N. Socci for helpful discussions. Work at Berkeley was supported in part by the National Science Foundation through a Presidential Young Investigator Award (to WB), supplemented by funds from Sun Microsystems and Cray Research, and by the Fannie and John Hertz Foundation through a graduate fellowship (to DLR). Work in Santa Barbara was supported in part by the NSF through Grant No. PHY82-17853, supplemented by funds from NASA.

## References

[1] W. Bialek. In E. Jen, editor, *1989 Lectures in Complex Systems, SFI Studies in the Sciences of Complexity, Lect. Vol. II*, pages 513–595. Addison-Wesley, Menlo Park, CA, 1990.

[2] H. B. Barlow. In W. A. Rosenblith, editor, *Sensory Communication*, page 217. MIT Press, Cambridge, MA, 1961.

[3] C. E. Shannon and W. Weaver. *The Mathematical Theory of Communication*. University of Illinois Press, Urbana, IL, 1949.

[4] D. Field. *J. Opt. Soc. Am.*, 4:2379, 1987.

[5] M. A. Webster and R. L. DeValois. *J. Opt. Soc. Am.*, 2:1124–1132, 1985.

[6] B. Sakitt and H. B. Barlow. *Biol. Cybern.*, 43:97–108, 1982.

[7] R. Linsker. In D. Touretzky, editor, *Advances in Neural Information Processing 1*, page 186. Morgan Kaufmann, San Mateo, CA, 1989.

[8] J. J. Atick and A. N. Redlich. *Neural Computation*, 2:308, 1990